# Doubly Stochastic Normalization for Spectral Clustering

**Ron Zass** and **Amnon Shashua** [*]

## Abstract

In this paper we focus on the issue of normalization of the affinity matrix in spectral clustering. We show that the difference between N-cuts and Ratio-cuts is in the error measure being used (relative-entropy versus $L_1$ norm) in finding the closest doubly-stochastic matrix to the input affinity matrix. We then develop a scheme for finding the optimal, under Frobenius norm, doubly-stochastic approximation using Von-Neumann's successive projections lemma. The new normalization scheme is simple and efficient and provides superior clustering performance over many of the standardized tests.

## 1 Introduction

The problem of partitioning data points into a number of distinct sets, known as the clustering problem, is central in data analysis and machine learning. Typically, a graph-theoretic approach to clustering starts with a measure of pairwise affinity $K_{ij}$ measuring the degree of similarity between points $\mathbf{x}_i, \mathbf{x}_j$, followed by a normalization step, followed by the extraction of the leading eigenvectors which form an embedded coordinate system from which the partitioning is readily available. In this domain there are three principle dimensions which make a successful clustering: (i) the affinity measure, (ii) the normalization of the affinity matrix, and (iii) the particular clustering algorithm. Common practice indicates that the former two are largely responsible for the performance whereas the particulars of the clustering process itself have a relatively smaller impact on the performance.

In this paper we focus on the normalization of the affinity matrix. We first show that the existing popular methods Ratio-cut (cf. [1]) and Normalized-cut [7] employ an implicit normalization which corresponds to $L_1$ and Relative Entropy based approximations of the affinity matrix $K$ to a doubly stochastic matrix. We then introduce a Frobenius norm ($L_2$) normalization algorithm based on a simple successive projections scheme (based on Von-Neumann's [5] successive projection lemma for finding the closest intersection of sub-spaces) which finds the closest doubly stochastic matrix under the least-squares error norm. We demonstrate the impact of the various normalization schemes on a large variety of data sets and show that the new normalization algorithm often induces a significant performance boost in standardized tests. Taken together, we introduce a new tuning dimension to clustering algorithms allowing better control of the clustering performance.

## 2 The Role of Doubly Stochastic Normalization

It has been shown in the past [11, 4] that K-means and spectral clustering are intimately related where in particular [11] shows that the popular affinity matrix normalization such as employed by Normalized-cuts is related to a doubly-stochastic constraint induced by K-means. Since this background is a key to our work we will briefly introduce the relevant arguments and derivations.

Let $\mathbf{x}_i \in R^N$, $i = 1, ..., n$, be points arranged in $k$ (mutually exclusive) clusters $\psi_1, .., \psi_k$ with $n_j$ points in cluster $\psi_j$ and $\sum_j n_j = n$. Let $K_{ij} = \kappa(\mathbf{x}_i, \mathbf{x}_j)$ be a symmetric positive-semi-definite

---

[*] School of Engineering and Computer Science, Hebrew University of Jerusalem, Jerusalem 91904, Israel.

affinity function, e.g. $K_{ij} = \exp^{-\|\mathbf{x}_i - \mathbf{x}_j\|^2/\sigma^2}$. Then, the problem of finding the cluster assignments by maximizing:

$$\max_{\psi_1,...,\psi_k} \sum_{j=1}^{k} \frac{1}{n_j} \sum_{(r,s)\in\psi_j} K_{r,s}, \tag{1}$$

is equivalent to minimizing the "kernel K-means" problem:

$$\min_{\mathbf{c}_1,...,\mathbf{c}_k\psi_1,...,\psi_k} \sum_{j=1}^{k} \sum_{i\in\psi_j} \|\phi(\mathbf{x}_i) - \mathbf{c}_j\|^2,$$

where $\phi(\mathbf{x}_i)$ is a mapping associated with the kernel $\kappa(\mathbf{x}_i, \mathbf{x}_j) = \phi(\mathbf{x}_i)^\top \phi(\mathbf{x}_j)$ and $\mathbf{c}_j = (1/n_j) \sum_{i\in\psi_j} \phi(\mathbf{x}_i)$ are the class centers. After some algebraic manipulations it can be shown that the optimization setup of eqn. 1 is equivalent to the matrix form:

$$\max_{G} tr(G^\top K G) \; s.t \;\; G \geq 0, \; GG^\top \mathbf{1} = \mathbf{1}, \; G^\top G = I \tag{2}$$

where $G$ is the desired assignment matrix with $G_{ij} = 1/\sqrt{n_j}$ if $i \in \psi_j$ and zero otherwise, and $\mathbf{1}$ is a column vector of ones. Note that the feasible set of matrices satisfying the constraints $G \geq 0, GG^\top \mathbf{1} = \mathbf{1}, G^\top G = I$ are of this form for some partitioning $\psi_1, ..., \psi_k$. Note also that the matrix $F = GG^\top$ must be doubly stochastic ($F$ is non-negative, symmetric and $F\mathbf{1} = \mathbf{1}$).

Taken together, we see that the desire is to find a doubly-stochastic matrix $F$ as close as possible to the input matrix $K$ (in the sense that $\sum_{ij} F_{ij} K_{ij}$ is maximized over all feasible $F$), such that the symmetric decomposition $F = GG^\top$ satisfies non-negativity ($G \geq 0$) and orthonormality constraints ($G^\top G = I$).

To see the connection with spectral clustering, and N-cuts in particular, relax the non-negativity condition of eqn. 2 and define a two-stage approach: find the closest doubly stochastic matrix $K'$ to $K$ and we are left with a spectral decomposition problem:

$$\max_{G} tr(G^\top K' G) \; s.t \;\; G^\top G = I \tag{3}$$

where $G$ contains the leading $k$ eigenvectors of $K'$. We will refer to the process of transforming $K$ to $K'$ as a *normalization* step. In N-cuts, the normalization takes the form $K' = D^{-1/2}KD^{-1/2}$ where $D = diag(K\mathbf{1})$ (a diagonal matrix containing the row sums of $K$) [9]. In [11] it was shown that repeating the N-cuts normalization, i.e., setting up the iterative step $K^{(t+1)} = D^{-1/2}K^{(t)}D^{-1/2}$ where $D = diag(K^{(t)}\mathbf{1})$ and $K^{(0)} = K$ converges to a doubly-stochastic matrix (a symmetric version of the well known "iterative proportional fitting procedure" [8]).

The conclusion of this brief background is to highlight the motivation for seeking a doubly-stochastic approximation to the input affinity matrix as part of the clustering process. The open issue is under what *error measure* is the approximation to take place? It is not difficult to show that repeating the N-cuts normalization converges to the global optimum under the *relative entropy* measure (see Appendix). Noting that spectral clustering optimizes the Frobenius norm it seems less natural to have the normalization step optimize a relative entropy error measure.

We will derive in this paper the normalization under the $L_1$ norm and under the Frobenius norm. The purpose of the $L_1$ norm is to show that the resulting scheme is equivalent to a ratio-cut clustering — thereby not introducing a new clustering scheme but only contributing to the unification and better understanding the differences between the N-cuts and Ratio-cuts schemes. The Frobenius norm normalization is a new formulation and is based on a simple iterative scheme. The resulting normalization provides a new clustering performance which proves quite practical and boosts the clustering performance in many of the standardized tests we conducted.

## 3 Ratio-cut and the $L_1$ Normalization

Given that our desire is to find a doubly stochastic approximation $K'$ to the input affinity matrix $K$, we begin with the $L_1$ norm approximation:

**Proposition 1 (ratio-cut)** *The closest doubly stochastic matrix $K'$ under the $L_1$ error norm is*

$$K' = K - D + I,$$

*which leads to the ratio-cut clustering algorithm, i.e., the partitioning of the data set into two clusters is determined by the second smallest eigenvector of the Laplacian $D - K$, where $D = diag(K\mathbf{1})$.*

**Proof:** Let $r = \min_F \|K - F\|_1$ $s.t.$ $F\mathbf{1} = \mathbf{1}$, $F = F^\top$, where $\|A\|_1 = \sum_{ij} abs(A_{ij})$ is the $L_1$ norm. Since $\|K - F\|_1 \geq \|(K - F)\mathbf{1}\|_1$ for any matrix $F$, we must have:

$$r \geq \|(K - F)\mathbf{1}\|_1 = \|D\mathbf{1} - \mathbf{1}\|_1 = \|D - I\|_1.$$

Let $F = K - D + I$, then

$$\|K - (K - D + I)\|_1 = \|D - I\|_1.$$

If $\mathbf{v}$ is an eigenvector of the Laplacian $D - K$ with eigenvalue $\lambda$, then $\mathbf{v}$ is also an eigenvector of $K' = K - D + I$ with eigenvalue $1 - \lambda$ and since $(D - K)\mathbf{1} = 0$ then the smallest eigenvector $\mathbf{v} = \mathbf{1}$ of the Laplacian is the largest of $K'$, and the second smallest eigenvector of the Laplacian (the ratio-cut result) corresponds to the second largest eigenvector of $K'$. $\Box$

What we have so far is that the difference between N-cuts and Ratio-cuts as two popular spectral clustering schemes is that the former uses the relative entropy error measure in finding a doubly stochastic approximation to $K$ and the latter uses the $L_1$ norm error measure (which turns out to be the negative Laplacian with an added identity matrix).

# 4   Normalizing under Frobenius Norm

Given that spectral clustering optimizes the Frobenius norm, there is a strong argument in favor of finding a Frobenius-norm optimum doubly stochastic approximation to $K$. The optimization setup is that of a quadratic linear programming (QLP). However, the special circumstances of our problem render the solution to the QLP to consist of a very simple iterative computation, as described next.

The closest doubly-stochastic matrix $K'$ under Frobenius norm is the solution to the following QLP:

$$K' = \text{argmin}_F \|K - F\|_F^2 \;\; s.t. \;\; F \geq 0, \; F\mathbf{1} = \mathbf{1}, \; F = F^\top, \tag{4}$$

where $\|A\|_F^2 = \sum_{ij} A_{ij}^2$ is the Frobenius norm. We define next two sub-problems, each with a closed-form solution, and have our QLP solution derived by alternating successively between the two until convergence. Consider the affine sub-problem:

$$P_1(X) = \text{argmin}_F \|X - F\|_F^2 \;\; s.t. \;\; F\mathbf{1} = \mathbf{1}, \; F = F^\top \tag{5}$$

and the convex sub-problem:

$$P_2(X) = \text{argmin}_F \|X - F\|_F^2 \;\; s.t. \;\; F \geq 0 \tag{6}$$

We will use the Von-Neumann [5] successive projection lemma stating that $P_1 P_2 P_1 P_2 ... P_1(K)$ will converge onto the projection of $K$ onto the intersection of the affine and conic subspaces described above[1]. Therefore, what remains to show is that the projections $P_1$ and $P_2$ can be solved efficiently (and in closed form).

We begin with the solution for $P_1$. The Lagrangian corresponding to eqn. 5 takes the form:

$$L(F, \mu_1, \mu_2) = trace(F^\top F - 2X^\top F) - \mu_1^\top (F\mathbf{1} - \mathbf{1}) - \mu_2^\top (F^\top \mathbf{1} - \mathbf{1}),$$

where from the condition $F = F^\top$ we have that $\mu_1 = \mu_2 = \mu$. Setting the derivative with respect to $F$ to zero yields:

$$F = X + \mu \mathbf{1}^\top + \mathbf{1}\mu^\top.$$

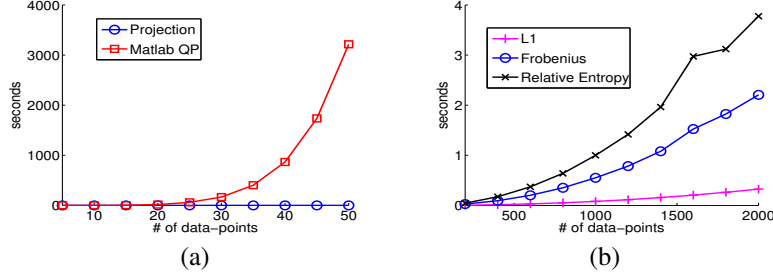

(a)    (b)

Figure 1: Running times of the normalization algorithms. (a) the Frobenius scheme compared to a general Matlab QLP solver, (b) running time of the three normalization schemes.

Isolate $\mu$ by multiplying by $\mathbf{1}$ on both sides: $\mu = (nI + \mathbf{1}\mathbf{1}^\top)^{-1}(I - X)\mathbf{1}$. Noting that $(nI + \mathbf{1}\mathbf{1}^\top)^{-1} = (1/n)(I - (1/2n)\mathbf{1}\mathbf{1}^\top)$ we obtain a closed form solution:

$$P_1(X) = X + \left(\frac{1}{n}I + \frac{\mathbf{1}^\top X \mathbf{1}}{n^2}I - \frac{1}{n}X\right)\mathbf{1}\mathbf{1}^\top - \frac{1}{n}\mathbf{1}\mathbf{1}^\top X. \tag{7}$$

The projection $P_2(X)$ can also be described in a simple closed form manner. Let $I_+$ be the set of indices corresponding to non-negative entries of $X$ and $I_-$ the set of negative entries of $X$. The criterion function $\|X - F\|_F^2$ becomes:

$$\|X - F\|_F^2 = \sum_{(i,j)\in I_+} (X_{ij} - F_{ij})^2 + \sum_{(i,j)\in I_-} (X_{ij} - F_{ij})^2.$$

Clearly, the minimum energy over $F \geq 0$ is obtained when $F_{ij} = X_{ij}$ for all $(i,j) \in I_+$ and zero otherwise. Let $th_{\geq 0}(X)$ stand for the operator that zeroes out all negative entries of $X$. Then,

$$P_2(X) = th_{\geq 0}(X).$$

To conclude, the global optimum of eqn. 4 which returns the closest doubly stochastic matrix $K'$ in Frobenius error norm to the input affinity matrix $K$ is obtained by repeating the following steps:

**Algorithm 1 (Frobenius-optimal Doubly Stochastic Normalization)** *finds the closest doubly stochastic approximation in Frobenius error norm to a given matrix $K$ (global optimum of eqn. 4).*

1. *Let $X^{(0)} = K$.*

2. *Repeat $t = 0, 1, 2, ...$*

    *(a) $X^{(t+1)} = P_1(X^{(t)})$*

    *(b) If $X^{(t+1)} \geq 0$ then stop and set $K' = X^{(t+1)}$, otherwise set $X^{(t+1)} = th_{\geq 0}(X^{(t+1)})$.*

This algorithm is simple and very efficient. Fig. 1a shows the running time of the algorithm compared to an off-the-shelf QLP Matlab solver over random matrices of increasing size — one can see that the run-time of our algorithm is a fraction of the standard QLP solver and scales very well with dimension. In fact the standard QLP solver can handle only small problem sizes. In Fig. 1b we plot the running times of all three normalization schemes: the $L_1$ norm (computing the Laplacian), the relative-entropy (the iterative $D^{-1/2}KD^{-1/2}$), and the Frobenius scheme presented in this section. The Frobenius is more efficient than the relative-entropy normalization (which is the least efficient among the three).

## 5   Experiments

For the clustering algorithm into $k \geq 2$ clusters we experimented with the spectral algorithms described in [10] and [6]. The latter uses the N-cuts normalization $D^{-1/2}KD^{-1/2}$ followed by K-means on the embedded coordinates (the leading $k$ eigenvectors of the normalized affinity) and

the former uses a certain discretization scheme to turn the $k$ leading eigenvectors into an indicator matrix. Both algorithms produced similar results thus we focused on [10] while replacing the normalization with the three schemes presented above. We refer to "Ncuts" as the original normalization $D^{-1/2}KD^{-1/2}$, by "RE" to the iterative application of the original normalization (which is proven to converge to a doubly stochastic matrix [11]), by "L1" to the $L_1$ doubly-stochastic normalization (which we have shown is equivalent to Ratio-cuts) and by "Frobenius" to the iterative Frobenius scheme based on Von-Neumann's lemma described in Section 4. We also included a "None" field which corresponds to no normalization being applied.

| Dataset | Kernel | k | Size | Dim. | Lowest Error Rate | | | | |
|---------|--------|---|------|------|------|-----------|------|-------|------|
| | | | | | L1 | Frobenius | RE | NCuts | None |
| SPECTF heart | RBF | 2 | 267 | 44 | 27.5 | **19.2** | 27.5 | 27.5 | 29.5 |
| Pima | RBF | 2 | 768 | 8 | 36.2 | 35.2 | **34.9** | 35.2 | 35.4 |
| Wine | RBF | 3 | 178 | 13 | 38.8 | **27.0** | 34.3 | 29.2 | 27.5 |
| SpamBase | RBF | 2 | 4601 | 57 | 36.1 | **30.3** | 37.7 | 31.8 | 30.4 |
| BUPA | Poly | 2 | 345 | 6 | **37.4** | **37.4** | 41.7 | 41.7 | **37.4** |
| WDBC | Poly | 2 | 569 | 30 | 18.8 | **11.1** | 37.4 | 37.4 | 18.8 |

Table 1: UCI datasets used, together with some characteristics and the best result achieved using the different methods.

| Dataset | Kernel | k | Size | #PC | Lowest Error Rate | | | | |
|---------|--------|---|------|-----|------|-----------|------|-------|------|
| | | | | | L1 | Frobenius | RE | NCuts | None |
| Leukemia | Poly | 2 | 72 | 5 | 27.8 | **16.7** | 36.1 | 38.9 | 30.6 |
| Lung | Poly | 2 | 181 | 5 | 15.5 | **9.9** | 16.6 | 15.5 | 15.5 |
| Prostate | RBF | 2 | 136 | 5 | 40.4 | **19.9** | 43.4 | 40.4 | 40.4 |
| Prostate Outcome | RBF | 2 | 21 | 5 | 28.6 | **4.8** | 23.8 | 28.6 | 28.6 |

Table 2: Cancer datasets used, together with some characteristics and the best result achieved using the different methods.

We begin with evaluating the clustering quality obtained under the different normalization methods taken over a number of well studied datasets from the UCI repository[2]. The data-sets are listed in Table 1 together with some of their characteristics. The best performance (lowest error rate) is presented in Boldface. With the first four datasets we used an RBF kernel $e^{\frac{\|x_i - x_j\|^2}{\sigma^2}}$ for the affinity matrix, while for the latter two a polynomial kernel $(x_i^T x_j + 1)^d$ was used. The kernel parameters were calibrated independently for each method and for each dataset. In most cases the best performance was obtained with the Frobenius norm approximation, but as a general rule the type of normalization depends on the data. Also worth noting are instances, such as *Wine* and *SpamBase*, when the RE or Ncuts actually worsen the performance. In that case the RE performance is worse the Ncuts as the entire normalization direction is counter-productive. When RE outperforms None it also outperforms Ncuts (as can be expected since Ncuts is the first step in the iterative scheme of RE).

With regard to tuning the affinity measure, we show in Fig. 2 the clustering performance of each dataset under each normalization scheme under varying kernel setting ($\sigma$ and $d$ values). Generally, the performance of the Frobenius normalization behaves in a smoother manner and is more stable under varying kernel settings than the other normalization schemes.

Our next set of experiments was over some well studied cancer data-sets[3]. The data-sets are listed in Table 2 together with some of their characteristics. The column "#PC" refers to the number of principal components used in a PCA pre-processing for the purpose of dimensionality reduction prior to clustering. Note that better results can be achieved when using a more sophisticated pre-processing, but since the focus is on the performances of the clustering algorithms and not on the datasets, we prefer not to use the optimal pre-processing and leave the data noisy. The *AML/ALL*

[2]http://www.ics.uci.edu/~ mlearn/MLRepository.html
[3]All cancer datasets can be found at http://sdmc.i2r.a-star.edu.sg/rp/

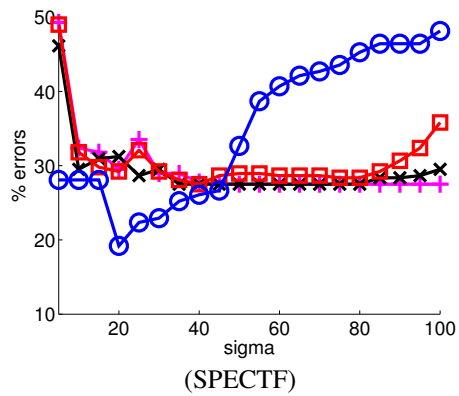

(SPECTF)

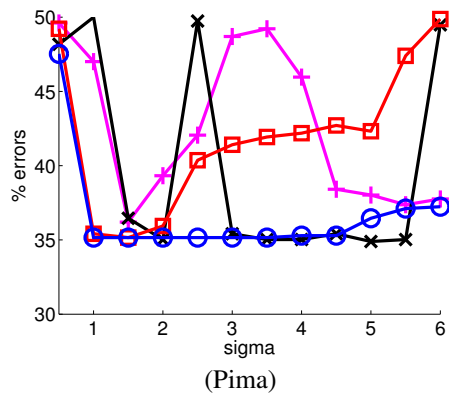

(Pima)

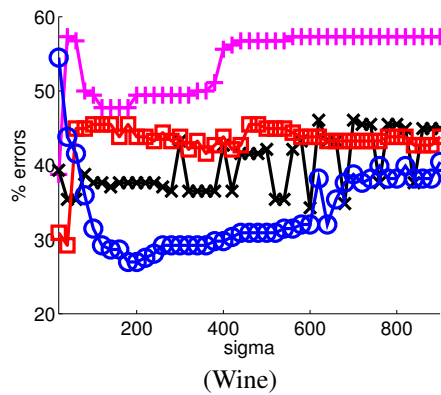

(Wine)

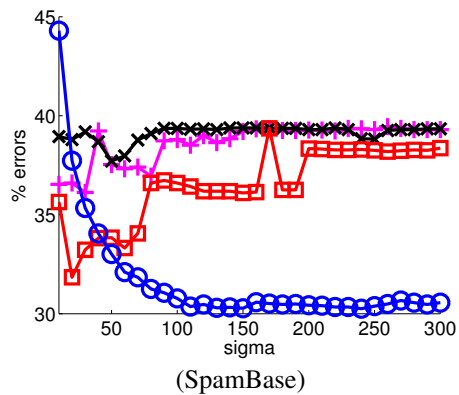

(SpamBase)

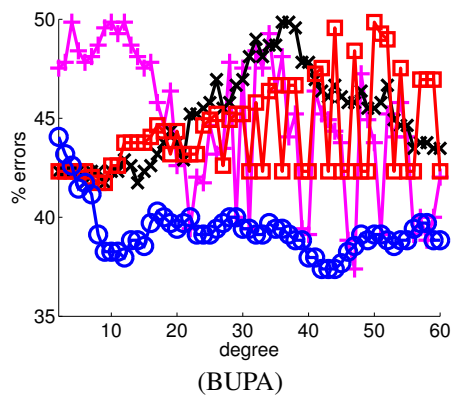

(BUPA)

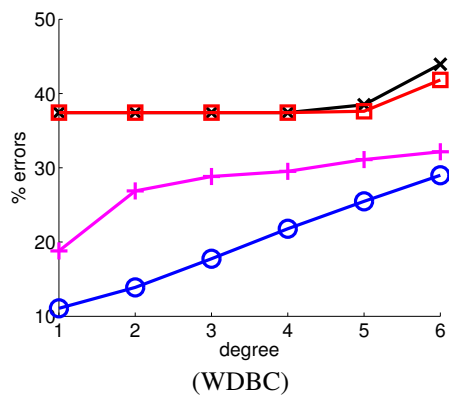

(WDBC)

Figure 2: Error rate vs. similarity measure, for the UCI datasets listed in Table 1
**L1** in magenta +; **Forbenius** in blue o; Relative Entropy in black ×; and Normalized-Cuts in red □

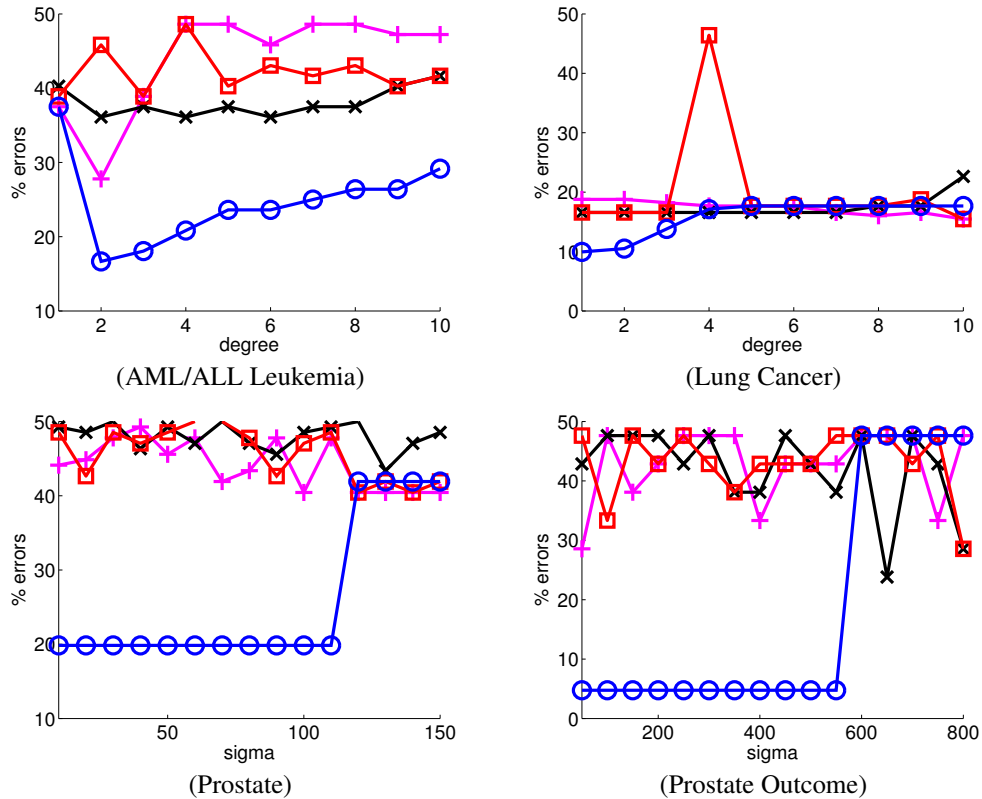

Figure 3: Error rate vs. similarity measure, for the cancer datasets listed in Table 2.
**L1** in magenta +; **Forbenius** in blue o; Relative Entropy in black ×; and Normalized-Cuts in red ▫

*Leukemia* dataset is a challenging benchmark common in the cancer community, where the task is to distinguish between two types of Leukemia. The original dataset consists of 7129 coordinates probed from 6817 human genes, and we perform PCA to obtain 5 leading principal components prior to clustering using a polynomial kernel. *Lung Cancer* (Brigham and Women's Hospital, Harvard Medical School) dataset is another common benchmark that describes 12533 genes sampled from 181 tissues. The task is to distinguish between malignant pleural mesothelioma (MPM) and adenocarcinoma (ADCA) of the lung. The *Prostate* dataset consists of 12,600 coordinates representing different genes, where the task is to identify prostate samples as tumor or non-tumor. We use the first five principal components as input for clustering using an RBF kernel. The *Prostate Outcome* dataset uses the same genes from another set of prostate samples, where the task is to predict the clinical outcome (relapse or non-relapse for at least four years). Finally, Fig. 3 shows the clustering performance of each dataset under each normalization scheme under varying kernel settings ($\sigma$ and $d$ values).

## 6   Summary

Normalization of the affinity matrix is a crucial element in the success of spectral clustering. The type of normalization performed by N-cuts is a step towards a doubly-stochastic approximation of the affinity matrix under relative entropy [11]. In this paper we have extended the normalization via doubly-stochasticity in three ways: (i) we have shown that the difference between N-Cuts and Ratio-cuts is in the error measure used to find the closest doubly stochastic approximation to the input affinity matrix, (ii) we have introduced a new normalization scheme based on Frobenius norm approximation. The scheme involves a succession of simple computations, is very simple to implement and is efficient computation-wise, and (iii) throughout extensive experimentation on standard data-sets we have shown the importance of normalization to the performance of spectral clustering.

In the experiments we have conducted the Frobenius normalization had the upper-hand in most cases. We have also shown that the relative-entropy normalization is not always the right approach as in some data-sets the performance worsened after the relative-entropy but never worsened when the Frobenius normalization was applied.

## Footnotes

[1] actually, the Von-Neumann lemma applies only to linear subspaces. The extension to convex subspaces involves a "deflection" component described by Dykstra [3]. However, it is possible to show that for this specific problem the deflection component is redundant and the Von-Neumann lemma still applies.

## References

[1] P. K. Chan, M. D. F. Schlag, and J. Y. Zien. Spectral k-way ratio-cut partitioning and clustering. *IEEE Transactions on Computer-aided Design of Integrated Circuits and Systems*, 13(9):1088–1096, 1994.

[2] I. Csiszar. I-divergence geometry of probability distributions and minimization problems. *The Annals of Probability*, 3(1):146–158, 1975.

[3] R.L. Dykstra. An algorithm for restricted least squares regression. *J. of the Amer. Stat. Assoc.*, 78:837–842, 1983.

[4] I.S.Dhillon, Y.Guan, and B.Kulis. Kernel k-means, spectral clustering and normalized cuts. In *International Conference on Knowledge Discovery and Data Mining(KDD)*, pages 551–556, Aug. 2004.

[5] J. Von Neumann. *Functional Operators Vol. II.* Princeton University Press, 1950.

[6] A.Y. Ng, M.I. Jordan, and Y. Weiss. On spectral clustering: Analysis and an algorithm. In *Proceedings of the conference on Neural Information Processing Systems (NIPS)*, 2001.

[7] J. Shi and J. Malik. Normalized cuts and image segmentation. *IEEE Transactions on Pattern Analysis and Machine Intelligence*, 22(8), 2000.

[8] R. Sinkhorn and P. Knopp. Conerning non-negative matrices and doubly stochastic matrices. *Pacific J. Math.*, 21:343–348, 1967.

[9] Y. Weiss. Segmentation using eigenvectors: a unifying view. In *Proceedings of the IEEE Conference on Computer Vision and Pattern Recognition*, 1999.

[10] S.X. Yu and J. Shi. Multiclass spectral clustering. In *Proceedings of the International Conference on Computer Vision*, 2003.

[11] R. Zass and A. Shashua. A unifying approach to hard and probabilistic clustering. In *Proceedings of the International Conference on Computer Vision*, Beijing, China, Oct. 2005.

## A   Normalized Cuts and Relative Entropy Normalization

The following proposition is an extension (symmetric version) of the claim about the iterative proportional fitting procedure converging in relative entropy error measure [2]:

**Proposition 2** *The closest doubly-stochastic matrix $F$ under the relative-entropy error measure to a given symmetric matrix $K$, i.e., which minimizes:*

$$\min_F \ RE(F\|K) \ \ s.t. \ F \geq 0, \ F = F^\top, \ F\mathbf{1} = 1, \ F^\top\mathbf{1} = 1$$

*has the form $F = DKD$ for some (unique) diagonal matrix $D$.*

**Proof:** The Lagrangian of the problem is:

$$L() = \sum_{ij} f_{ij} \ln \frac{f_{ij}}{k_{ij}} + \sum_{ij} k_{ij} - \sum_{ij} f_{ij} - \sum_i \lambda_i (\sum_j f_{ij} - 1) - \sum_j \mu_j (\sum_i f_{ij} - 1)$$

The derivative with respect to $f_{ij}$ is:

$$\frac{\partial L}{\partial f_{ij}} = \ln f_{ij} + 1 - \ln k_{ij} - 1 - \lambda_i - \mu_j = 0$$

from which we obtain:

$$f_{ij} = e^{\lambda_i} e^{\mu_j} k_{ij}$$

Let $D_1 = diag(e^{\lambda_1}, ..., e^{\lambda_n})$ and $D_2 = diag(e^{\mu_1}, ..., e^{\mu_n})$, then we have:

$$F = D_1 K D_2$$

Since $F = F^\top$ and $K$ is symmetric we must have $D_1 = D_2$. □